# LTD Facilitates Learning In a Noisy Environment

**Paul Munro**
School of Information Sciences
University of Pittsburgh
Pittsburgh PA 15260
*pwm+@pitt.edu*

**Gerardina Hernandez**
Intelligent Systems Program
University of Pittsburgh
Pittsburgh PA 15260
*gehst5+@pitt.edu*

## Abstract

Long-term potentiation (LTP) has long been held as a biological substrate for associative learning. Recently, evidence has emerged that long-term depression (LTD) results when the presynaptic cell fires after the postsynaptic cell. The computational utility of LTD is explored here. Synaptic modification kernels for both LTP and LTD have been proposed by other laboratories based studies of one postsynaptic unit. Here, the interaction between time-dependent LTP and LTD is studied in small networks.

## 1 Introduction

Long term potentiation (LTP) is a neurophysiological phenomenon observed under laboratory conditions in which two neurons or neural populations are stimulated at a high frequency with a resulting measurable increase in synaptic efficacy between them that lasts for several hours or days [1]-[2]  LTP thus provides direct evidence supporting the neurophysiological hypothesis articulated by Hebb [3].

This increase in synaptic strength must be countered by a mechanism for weakening the synapse [4]. The biological correlate, long-term depression (LTD) has also been observed in the laboratory; that is, synapses are observed to weaken when low presynaptic activity coincides with high postsynaptic activity [5]-[6].

Mathematical formulations of Hebbian learning produce weights, $w_{ij}$, (where $i$ is the presynaptic unit and $j$ is the postsynaptic unit), that capture the covariance [Eq. 1] between the instantaneous activities of pairs of units, $a_i$ and $a_j$ [7].

$$\dot{w}_{ij}(t) = (a_i(t) - \overline{a}_i)(a_j(t) - \overline{a}_j) \qquad [1]$$

This idea has been generalized to capture covariance between activities that are shifted in time [8]-[9], resulting in a framework that can model systems with temporal delays and dependencies [Eq. 2].

$$\dot{w}_{ij}(t) = \iint K(t'' - t') a_i(t'') a_j(t') dt'' dt' \qquad [2]$$

As will be shown in the following sections, depending on the choice of the function $K(\Delta t)$, this formulation encompasses a broad range of learning rules [10]-[12] and can support a comparably broad range of biological evidence.

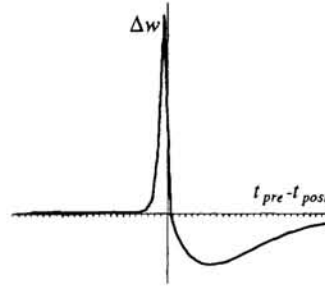

Figure 1. Synaptic change as a function of the time difference between spikes from the presynaptic neuron and the postsynaptic neuron. Note that for $t_{pre} < t_{post}$, LTP results ($\Delta w > 0$), and for $t_{pre} > t_{post}$, the result is LTD.

Recent biological data from [13]-[15], indicates an increase in synaptic strength (LTP) when presynaptic activity precedes postsynaptic activity, and LTD in the reverse case (postsynaptic precedes presynaptic). These ideas have started to appear in some theoretical models of neural computation [10]-[12], [16]-[18]. Thus, Figure 1 shows the form of the dependence of synaptic change, $\Delta w$ on the difference in spike arrival times.

## 2 A General Framework

Given specific assumptions, the integral in Eq. 2 can be separated into two integrals, one representing LTP and one representing LTD [Eq. 3].

$$\dot{w}_{ij}(t) = \underbrace{\int_{t'=-\infty}^{t} K_P(t-t')a_i(t')a_j(.)dt'}_{LTP} + \underbrace{\int_{t'=-\infty}^{t} K_D(t-t')a_i(t)a_j(t')dt'}_{LTD} \qquad [3]$$

The activities that do not depend on $t'$ can be factored out of the integrals, giving two Hebb-like products, between the instantaneous activity in one cell and a weighted time average of the activity in the other [Eq. 4]:

$$\dot{w}_{ij}(t) = \left\langle a_i(t) \right\rangle_P a_j(t) - a_i(t) \left\langle a_j(t) \right\rangle_D$$

$$\text{where } \left\langle f(t) \right\rangle_X \equiv | \int_{t'=-\infty}^{t} K_X(t-t')f(t')dt' | \text{ for } X \in \{P,D\} \qquad [4]$$

The kernel functions $K_P$ and $K_D$ can be chosen to select precise times out of the convoluted function $f(t)$, or to average across the functions for an arbitrary range. The alpha function is useful here [Eq. 5]. A high value of $\alpha$ selects an immediate time, while a small value approximates a longer time-average.

$$K_X(\tau) = \beta_X \tau e^{-\alpha_X \tau} \text{ for } X \in \{P,D\}$$
$$\text{with } \alpha_P > 0, \alpha_D > 0, \beta_P > 0, \beta_D < 0 \qquad [5]$$

For high values of $\alpha_P$ and $\alpha_D$, only pre- and post- synaptic activities that are very close temporally will interact to modify the synapse. In a simulation with discrete step sizes, this can be reasonably approximated by only considering just a single time step [Eq. 6].

$$\Delta w_{ij}(t) = a_i(t-1)a_j(t) - a_i(t)a_j(t-1) \tag{6}$$

Summing $\Delta w_{ij}(t)$ and $\Delta w_{ij}(t+1)$ gives a net change in the weights $\Delta^{(2)}w_{ij} = w_{ij}(t+1)-w_{ij}(t-1)$ over the two time steps:

$$\Delta^{(2)}w_{ij}(t) = a_i(t)\Delta^{(2)}a_j(t) - a_j(t)\Delta^{(2)}a_i(t) \tag{7}$$

The first term is predictive in that it has the form of the delta rule where $a_j(t+1)$ acts as a training signal for $a_j(t-1)$, as in a temporal Hopfield network [9].

## 3   Temporal Contrast Enhancement

The computational role of the LTP term in Eq. 3 is well established, but how does the second term contribute? A possibility is that the term is analogous to lateral inhibition in the temporal domain; that is, that by suppressing associations in the "wrong" temporal direction, the system may be more robust against noise in the input. The resulting system may be able to detect the onset and offset of a signal more reliably than a system not using an anti-Hebbian LTD term.

The extent to which the LTD term is able to enhance temporal contrast is likely to depend idiosyncratically on the statistical qualities of a particular system. If so, the parameters of the system might only be valid for signals with specific statistical properties, or the parameters might be adaptive. Either of these possibilities lies beyond the scope of analysis for this paper.

## 4   Simulations

Two preliminary simulation studies illustrate the use of the learning rule for predictive behavior and for temporal contrast enhancement. For every simulation, kernel functions were specified by the parameters $\alpha$ and $\beta$, and the number of time steps, $n_P$ and $n_D$, that were sampled for the approximation of each integral.

### 4.1   Task 1.  A Sequential Shifter

The first task is a simple shifter over a set of 7 to 20 units. The system is trained on these stimuli and then tested to see if it can reconstruct the sequence given the initial input. The task is given with no noise and with temporal noise (see Figure 2). Task 1 is designed to examine the utility of LTD as an approach to learning a sequence with temporal noise. The ability of the network to reconstruct the noise-free sequence after training on the noisy sequence was tested for different LTD kernel functions.

Note that the same patterns are presented (for each time slice, just one of the $n$ units is active), but the shifts either skip or repeat *in time*. Experiments were run with $k = 1, 2,$ or 3 of the units active.

### 4.2   Task 2.  Time series reconstruction.

In this task, a set of units was trained on external sinusoidal signals that varied according to frequency and phase. The purpose of this task is to examine the role of LTD in providing temporal context. The network was then tested under a condition in which the

external signals were provided to all but one of the units. The activity of the deprived unit was then compared with its training signal

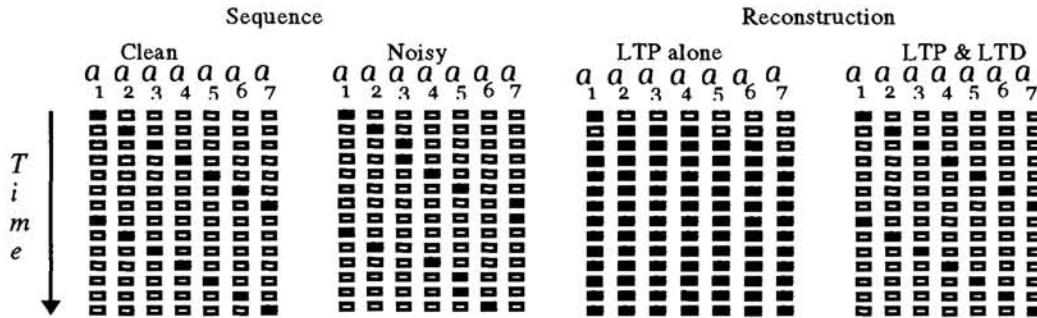

Figure 2. Reconstruction of clean shifter sequence using as input the noisy stimulus shifter sequence. For each time slice, just one of the 7 units is active. In the clean sequence, activity shifts cyclically around the 7 units. The noisy sequence has a random jitter of ±1.

## 5   ResultsSequential Shifter Results

All networks trained on the clean sequence can learn the task with LTP alone, but no networks could learn the shifter task based on a noisy training sequence unless there was also an LTD term. Without an LTD term, most units would saturate to maximum values. For a range of LTD parameters, the network would converge without saturating. Reconstruction performance was found to be sensitive to the LTD parameters. The parameters $\alpha$ and $\beta$ shown in Table.1 needed to be chosen very specifically to get perfect reconstruction (this was done by trial and error). For a narrow range of parameters near the optimal values, the reconstructed sequence was close to the noise-free target. However, the parameters $\alpha$ and $\beta$ shown in Table 2 are estimated from the experimental result of Zhang,et al [15].

Table 1.  Results of the sequential shifter task.

| $k$ | $n$ | $n_r$ | $\alpha_P$ | $\beta_P$ | $n_P$ | $\alpha_D$ | $\beta_D$ | $n_D$ | Time |
|---|---|---|---|---|---|---|---|---|---|
| 1 | 7 | 1 | 1 | 2.72 | 1 | 0.1 | -0.4 | 5 | 208 |
| 2 | 7 | 1 | 1 | 2.72 | 1 | 0.1 | -0.4 | 4 | 40 |
|   |   | 2 | 0.5 | 0.4 | 3 | 0.2 | -0.1 | 7 | 192 |
| 3 | 7 | 1 | 0.5 | 0.4 | 1 | 0.2 | -0.1 | 6 | 168 |
| 1 | 10 | 1 | 1 | 2.72 | 1 | 0.1 | -0.4 | 8 | 682 |
| 2 | 10 | 1 | 1 | 2.72 | 1 | 0.1 | -0.4 | 7 | 99 |
| 1 | 15 | 1 | 1 | 2.72 | 1 | 0.1 | -0.4 | 13 | 1136 |
| 1 | 20 | 1 | 1 | 2.72 | 1 | 0.1 | -0.4 | 18 | 4000 |

The task was to shift a pattern 1 unit with each time step. A block of $k$ of $n$ units was active. The parameters of the kernel functions ($\alpha$ and $\beta$), the number of values sampled from the kernel (the number of time slices used to estimate the integral), $n_P$ and $n_D$, and the number of steps used to begin the reconstruction, $n_r$ (usually $n_r = 1$) are given in the table. The last column of the table (*Time*) reports the number of iterations required for perfect reconstruction.

Table 2. Results of the sequential shifter task using as parameters: $n_r = 1$; $n_P = 1$;
$\alpha_D = 0.125$; $\alpha_P = 0.5$; $\beta_D = -\alpha_D * e * 0.35$; $\beta_P = \alpha_P * e * 0.8$.

| k | n | $n_D$ | Time |
|---|---|-------|------|
| 1 | 7 | 6 | 288 |
| 2 | 7 | 5 | 96 |
| 3 | 7 | 4 | 64 |

For the above results, the $k$ active units were always adjacent with respect to the shifting direction. For cases with noncontiguous active units, reconstruction was never exact. Networks trained with LTP alone would saturate, but would converge to a sequence "close" to the target (Fig. 3) if an LTD term was added.

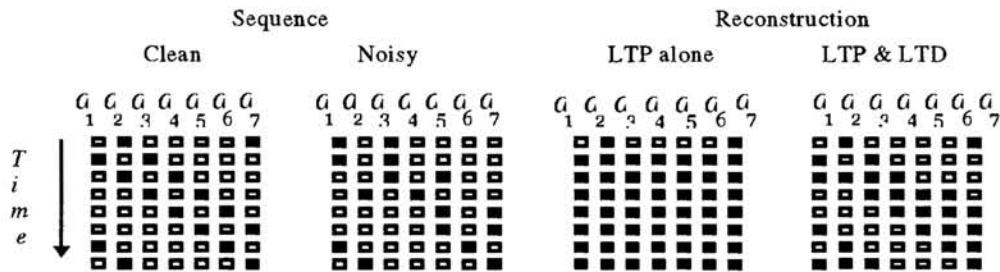

Figure 3. This base pattern ($k=2$, $n=7$) with noncontiguous active units was presented as a shifted sequence with noise. The target sequence is partially reconstructed only when LTP and LTD are used together.

## 5.1 Time Series Reconstruction Results

A network of just four units was trained for hundreds of iterations, the units were each externally driven by a sinusoidally varying input. Networks trained with LTP alone fail to reconstruct the time series on units deprived of external input during testing. In these simulations, there is no noise in the patterns, but LTD is shown to be necessary for reconstruction of the patterns (Fig. 4).

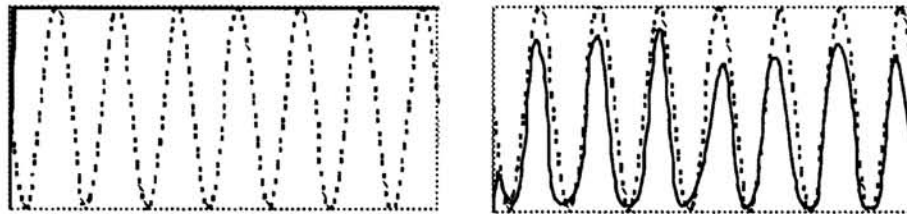

Figure 4. Reconstruction of sinusoids. Target signals from training (dashed) plotted with reconstructed signals (solid). Left: The best reconstruction using LTP alone. Right: A typical result with LTP and LTD together.

For high values of $\alpha_P$ and $\alpha_D$, the reconstruction of sinusoids is very sensitive to the values of $\beta_D$ and $\beta_P$. Figure 5 shows the results when $|\beta_D|$ and $\beta_P$ values are close. In the first case (top), when $|\beta_D|$ is slightly smaller than $\beta_P$, the first two neurons (from left to right) saturate. And, in the contrary case (bottom) the first two neurons

show almost null activation. However, the third and fourth neurons (from left to right) in both cases (top and bottom) show predictive behavior.

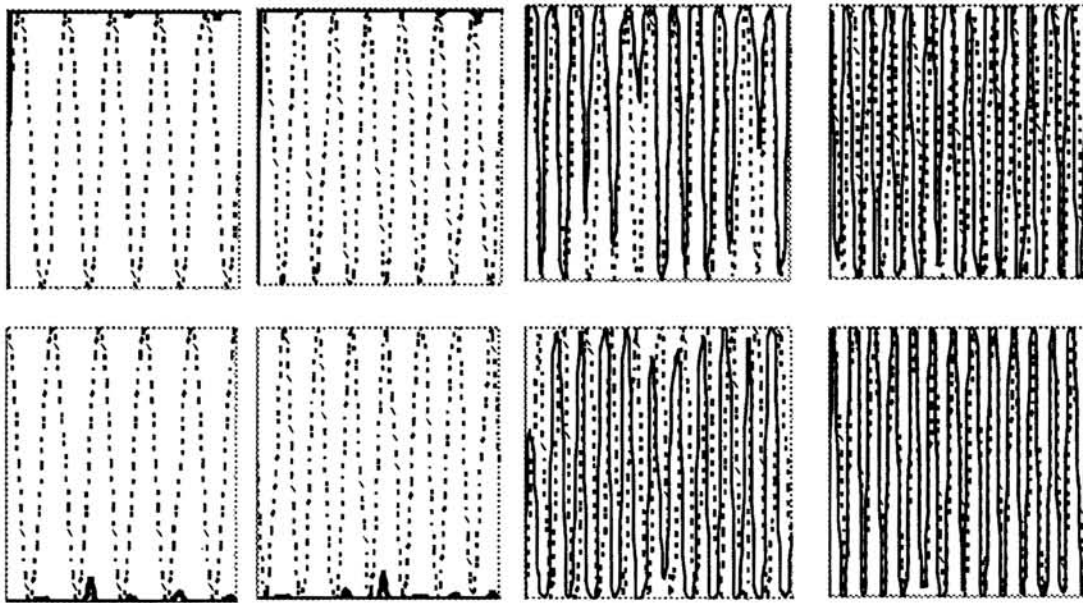

Figure 5. Reconstruction of sinusoids . Examples of target signals from training (dashed) plotted with reconstructed signals (solid). Top: When $|\beta_D|<\beta_P$ . Bottom: When $|\beta_D|>\beta_P$ .

## 6 Discussion

In the half century that has elapsed since Hebb articulated his neurophysiological postulate, the neuroscience community has come to recognize its fundamental role in plasticity. Hebb's hypothesis clearly transcends its original motivation to give a neurophysiologically based account of associative memory.

The phenomenon of LTP provides direct biological support for Hebb's postulate, and hence has clear cognitive implications. Initially after its discovery in the laboratory, the computational role of LTD was thought to be the flip side of LTP. This interpretation would have synapses strengthen when activities are correlated and have them weaken when they are anti-correlated. Such a theory is appealing for its elegance, and has formed the basis many network models [19]-[20]. However, the dependence of synaptic change on the relative timing of pre- and post- synaptic activity that has recently been shown in the laboratory is inconsistent with this story and calls for a computational interpretation. A network trained with such a learning rule cannot converge to a state where the weights are symmetric, for example, since $\Delta w_{ij} \neq \Delta w_{ji}$.

While the simulations reported here are simple and preliminary, they illustrate two tasks that benefit from the inclusion of time-dependent LTD. In the case of the sequential shifter, an examination of more complex predictive tasks is planned in the near future. It is expected that this will require architectures with unclamped (hidden) units. The role of LTD here is to temporally enhance contrast, in a way analogous to the role of lateral inhibition for computing spatial contrast enhancement in the retina. The time-series example illustrates the possible role of LTD for providing temporal context.

# 7  References

[1] Bliss TVP & Lφmo T (1973) Long-lasting potentiation of synaptic in the dentate area of the unanaesthetized rabbit following stimulation of the perforant path.*J Physiol* **232**:331-356

[2] Malenka RC (1995) LTP and LTD: dynamic and interactive processes of synaptic plasticity. *The Neuroscientist* 1:35-42.

[3] Hebb DO (1949) *The Organization of Behavior.* Wiley: NY.

[4] Stent G (1973) A physiological, mechanism for Hebb's postulate of learning. *Proc. Natl. Acad. Sci. USA* **70**: 997-1001

[5] Barrionuevo G, Schottler F & Lynch G (1980) The effects of repetitive low frequency stimulation on control and "pontentiated" synaptic responses in the hippocampus. *Life Sci* **27**:2385-2391.

[6] Thiels E, Xie X, Yeckel MF, Barrionuevo G & Berger TW (1996) NMDA Receptor-dependent LTD in different subfields of hippocampus in vivo and in vitro. *Hippocampus* **6**:43-51.

[7] Sejnowski T J (1977) Storing covariance with nonlinearly interacting neurons. *J. Math. Biol* 4:303-321.

[8] Sutton RS (1988) Learning to predict by the methods of temporal difference. *Machine Learning.* 3:9-44

[9] Sompolinsky H and Kanter I (1986) Temporal association in asymmetric neural networks. *Phys.Rev.Letter.* **57**:2861-2864.

[10] Gerstner W, Kempter R, van Hemmen JL & Wagner H (1996) A neuronal learning rule for sub-millisecond temporal coding. *Nature* **383**:76-78.

[11] Kempter R, Gerstner W & van Hemmen JL (1999) Spike-based compared to rate-based hebbian learning.Kearns, Ms., Solla, S.A and Cohn, D.A. Eds. *Advances in Neural Information Processing Systems 11.* MIT Press, Cambridge MA.

[12] Kempter R, Gerstner W, van Hemmen JL & Wagner H (1996) Temporal coding in the sub-millisecond range: Model of barn owl auditory pathway. Touretzky, D.S, Mozer, M.C, Hasselmo, M.E, Eds. *Advances in Neural Information Processing  Systems 8.* MIT Press, Cambridge MA pp.124-130.

[13] Markram H, Lubke J, Frotscher M & Sakmann B (1997) Regulation of synaptic efficacy by coincidence of postsynaptic Aps and EPPSPs. *Science* **275**:213-215.

[14] Markram H & Tsodyks MV (1996) Redistribution of synaptic efficacy between neocortical  pyramidal neurons. *Nature* **382**:807-810.

[15] Zhang L, Tao HW, Holt CE & Poo M (1998) A critical window for cooperation and competition among developing retinotectal synapses. *Nature* **35**:37-44

[16] Abbott LF, & Blum KI (1996) Functional significance of long-term potentiation for sequence learning and prediction. *Cerebral Cortex* **6**: 406-416.

[17] Abbott LF, & Song S (1999) Temporally asymmetric hebbian learning, spike timing and neuronal response variability. Kearns, Ms., Solla, S.A and Cohn, D.A. Eds. *Advances in Neural Information Processing Systems 11.* MIT Press, Cambridge MA.

[18] Goldman MS, Nelson SB & Abbott LF (1998) Decorrelation of spike trains by synaptic depression. *Neurocomputing* (in press).

[19] Hopfield J (1982) Neural networks and physical systems with emergent collective computational properties. *Proc. Natl. Acad. Sci. USA.* **79**:2554-2558.

[20] Ackley DH, Hinton GE, Sejnowski TJ (1985) A learning algorithm for Boltzmann machines. *Cognitive Science* **9**:147-169.
